# Dynamic Pruning of Factor Graphs for Maximum Marginal Prediction

**Christoph H. Lampert**
IST Austria (Institute of Science and Technology Austria)
Am Campus 1, 3400 Klosterneuburg, Austria
http://www.ist.ac.at/~chl    chl@ist.ac.at

## Abstract

We study the problem of maximum marginal prediction (MMP) in probabilistic graphical models, a task that occurs, for example, as the Bayes optimal decision rule under a Hamming loss. MMP is typically performed as a two-stage procedure: one estimates each variable's marginal probability and then forms a prediction from the states of maximal probability.

In this work we propose a simple yet effective technique for accelerating MMP when inference is sampling-based: instead of the above two-stage procedure we directly estimate the posterior probability of each decision variable. This allows us to identify the point of time when we are sufficiently certain about any individual decision. Whenever this is the case, we dynamically prune the variables we are confident about from the underlying factor graph. Consequently, at any time only samples of variables whose decision is still uncertain need to be created.

Experiments in two prototypical scenarios, multi-label classification and image inpainting, show that adaptive sampling can drastically accelerate MMP without sacrificing prediction accuracy.

## 1 Introduction

Probabilistic graphical models (PGMs) have become useful tools for classical machine learning tasks, such as multi-label classification [1] or semi-supervised learning [2], as well for many real-world applications, for example image processing [3], natural language processing [4], bioinformatics [5], and computational neuroscience [6]. Despite their popularity, the question of how to best perform (approximate) inference in any given graphical models is still far from solved. While variational approximations and related message passing algorithms have been proven useful for certain classes of models (see [7] for an overview), there is still a large number of cases for which sampling-based approaches are the safest choice. Unfortunately, inference by sampling is often computationally costly: many samples are required to reach a confident result, and generating the individual samples can be a complex task in itself, in particular if the underlying graphical model is large and highly connected.

In this work we study a particular inference problem: *maximum marginal prediction (MMP)* in binary-valued PGMs, i.e. the task of determining for each variable in the graphical model which of its states has highest marginal probability. MMP occurs naturally as the Bayes optimal decision rule under Hamming loss [8], and it has also found use as a building block for more complex prediction tasks, such as $M$-best MAP prediction [9]. The standard approach to sampling-based MMP is to estimate each variable's marginal probability distribution from a set of samples from the joint probability, and for each variable pick the state of highest estimated marginal probability. In this work, we propose an almost as simple, but more efficient way. We introduce one binary indicator variable for each decision we need to make, and keep estimates of the posterior probabilities of each of these during the process of sampling. As soon as we are confident enough about any of

the decisions, we remove it from the factor graph that underlies the sampling process, so no more samples are generated for it. Consequently, the factor graph shrinks over time, and later steps in the sampling procedure are accelerated, often drastically so.

Our main contribution lies in the combination of two relatively elementary components that we will introduce in the following section: an estimate for the posterior distributions of the decision variables, and a mean field-like construction for removing individual variables from a factor graph.

## 2   Adaptive Sampling for Maximum Marginal Prediction

Let $p(x)$ be a fixed probability distribution over the set $\mathcal{X} = \{0,1\}^V$ of binary labelings of a vertex set $V = \{1, \ldots, n\}$. We assume that $p$ is given to us by means of a factor graph, $\mathcal{G} = (V, \mathcal{F})$, with factor set $\mathcal{F} = \{F_1 \ldots, F_k\}$. Each factor, $F_j \subset V$, has an associated log-potential, $\psi_j$, which is a real-valued function of only the variables occurring in $F_j$. Writing $x_{F_j} = (x_i)_{i \in \mathcal{F}_j}$ we have

$$p(x) \propto \exp\left(-E(x)\right) \qquad \text{with } E(x) = \sum\nolimits_{F \in \mathcal{F}} \psi_F(x_F). \qquad (1)$$

for any $x \in \{0,1\}^V$. Our goal is *maximum marginal prediction*, i.e. to infer the values of decision variables $(z_i)_{i \in V}$ that are defined by $z_i := 0$ if $\mu_i \leq 0.5$, and $z_i := 1$ otherwise, where $\mu_i := p(x_i = 1)$ is the marginal probability of the $i$th variable taking the value 1. Computing the marginals $\mu_i$ in a loopy graphical model is in general #P-complete [10], so one has to settle for approximate marginals and approximate predictions. In this work, we assume access to a suitable constructed sampler based on the Markov chain Monte Carlo (MCMC) principle [11, 12], e.g. a Gibbs sampler [3] It produces a chain of states $\mathcal{S}_m = \{x^{(1)}, \ldots, x^{(N)}\}$, where each $x^{(i)}$ is a random sample from the joint distribution $p(x)$. From the set of sample we can compute an estimate, $\hat{\mu}_i = \frac{1}{m} \sum_{j=1}^m x_i^{(j)}$ of the true marginal, $\mu_i$, and make approximate decisions: $\hat{z}_i := 1$ if and only if $\hat{\mu}_i \geq 0.5$. Under mild conditions on the sampling procedure the law of large number guarantees that $\lim_{N \to \infty} \hat{\mu}_i = \mu_i$, and the decisions will become correct almost surely.

The main problem with sampling-based inference is when to stop sampling [13]. The more samples we have, the lower the variance on the estimates, so the more confident we can be about our decisions. However, each sample we generate increases the computational cost at least proportionally to the numbers of factors and variables involved. At the same time, the variance of the estimators $\hat{\mu}_i$ is reduced only proportionally to the square root of the sample size. In combination, this means that often, one spends a large amount of computational resources on a small win in predictive accuracy.

In the rest of this section, we explain our proposed idea of adaptive sampling in graphical models, which reduces the number of variables and factors during the course of the sampling procedure. As an illustrative example we start by the classical situation of adaptive sampling in the case of a single binary variable. This is a special case of Bayesian hypothesis selection, and –for the case of i.i.d. data– has recently also been rediscovered in the pattern recognition literature, for example for evaluating decision trees [14]. We then introduce our proposed extensions to correlated samples, and show how the per-variable decisions can be applied in the graphical model situation with potentially many variables and dependencies between them.

### 2.1   Adaptive Sampling of Binary Variables

Let $x$ be a single binary variable, for which we have a set of samples, $\mathcal{S} = \{x^{(1)}, \ldots, x^{(N)}\}$, available. The main insight lies in the fact that even though samples are used to empirically estimate the (marginal) probability $\mu$, the latter is not the actual quantity of interest to us. Ultimately, we are only interested in the value of the associated decision variable $z$.

**Independent samples.**   Assuming for the moment that the samples are independent (i.i.d.), we can derive an analytic expression for the posterior probability of $z$ given the observed samples,

$$p(z = 0|\mathcal{S}) = \int_0^{\frac{1}{2}} p(q|\mathcal{S})dq \qquad (2)$$

where $p(q|\mathcal{S})$ is the conditional probability density for $\mu$ having the value $q$. Applying Bayes' rule with likelihood $p(x|q) = q^x(1-q)^{1-x}$ and uniform prior, $p(q) = 1$, results in

$$= \frac{1}{B(m+1, N-m+1)} \int_0^{\frac{1}{2}} q^m (1-q)^{N-m} \, dq \quad = \quad I_{\frac{1}{2}}(m+1, N-m+1), \quad (3)$$

where $m = \sum_{j=1}^{N} x^{(j)}$. The normalization factor $B(\alpha, \beta) = \frac{\Gamma(\alpha)\Gamma(\beta)}{\Gamma(\alpha+\beta)}$ is the *beta function*; the integral is called the *incomplete beta function* (here evaluated at $\frac{1}{2}$). In combination, they form the *regularized incomplete beta function* $I_x(\alpha, \beta)$ [15].

From the above derivation we obtain a stopping criterion of $\epsilon$-*confidence*: given any number of samples we compute $p(z = 0|\mathcal{S})$ using Equation (3). If its value is above $1 - \epsilon$, we are $\epsilon$-confident that the correct decision is $z = 0$. If it is below $\epsilon$, we are equally confident that the correct decision is $z = 1$. Only if it lies inbetween we need to continue sampling. An analogue derivation to the above leads to a confidence bound for estimates of the marginal probability, $\hat{\mu} = m/N$, itself:

$$p(|\hat{\mu} - \mu| \leq \delta|\mathcal{S}) = I_{\hat{\mu}+\delta}(m+1, N-m+1) - I_{\hat{\mu}-\delta}(m+1, N-m+1). \quad (4)$$

Note that both tests are computable fast enough to be done after each sample, or small batches of samples. Evaluating the regularized incomplete beta function does not require numeric integration, and for fixed parameter $\epsilon$ the values $N$ and $m$ that bound the regions of confidence can also be tabulated [16]. A figure illustrating the difference between confidence in the MMP, and confidence in the estimated marginals can be found in the supplemental material. It shows that only relatively few independent samples (tens to hundreds) are sufficient to get a very confident MMP decision, if the actual marginals are close to 0 or 1. Intuitively, this makes sense, since in this situation a even coarse estimate of the marginal is sufficient to make of a decision with low error probability. Only if the true marginal lies inside of a relatively narrow interval around 0.5, the MMP decision becomes hard, and a large number of samples will be necessary to make a confident decision. Our experiments in Section 4 will show that in practical problem where the probability distribution is learned from data, the regions close to 0 and 1 are in fact the most relevant ones.

**Dependent samples.** Practical sampling procedures, such as MCMC, do not create i.i.d. samples, but dependent ones. Using the above bounds directly with these would make the tests overconfident. We overcome this problem, approximately, by borrowing the concept of *effective sample size (ESS)* from the statistics literature. Intuitively, the ESS reflects how many independent samples, $N'$, a set of $N$ correlated sample is equivalent to. In first order [1], one estimates the effective sample size as $N' = \frac{1-r}{1+r} N$, where $r$ is the first order autocorrelation coefficient, $r = \frac{1}{N-1} \sum_{j=1}^{N-1} \frac{(x^{(j)} - \hat{\mu})(x^{(j+1)} - \hat{\mu})}{\sigma^2}$, and $\sigma^2$ is the estimated variance of the sample sequence. Consequently, we can adjust the confidence tests defined above to correlated data: we first collecting a small number of samples, $N_0$, which we use to estimate initial values of $\sigma^2$ and $r$. Subsequently, we estimate the confidence of a decision by

$$p(z = 0|\mathcal{S}) = I_{\frac{1}{2}}(\hat{\mu}N' + 1, (1-\hat{\mu})N' + 1), \quad (5)$$

i.e. we replace the sample size $N$ by the effective sample size $N'$ and the raw count $m$ by its adjusted value $\hat{\mu}N'$.

## 2.2 Adaptive Sampling in Graphical Models

In this section we extend the above confidence criterion from single binary decisions to the situation of joint sampling from the joint probability of multiple binary variables. Note that we are only interested in per-variable decisions, so we can treat the value of each variable $x_i^{(j)}$ in a joint sample $x^{(j)}$ as a separate sample from the marginal probability $p(x_i)$. We will have to take the dependence between different samples $x_i^{(j)}$ and $x_i^{(k)}$ into account, but between variable dependencies within a sample do not pose problems. Consequently, estimate the confidence of any decision variable $z_i$ is straight forward from Equation (5), applied separately to the binary sample set $S_i = \{x_i^{(1)}, \ldots, x_i^{(N)}\}$. Note that all quantities defined above for the single variable case need to be computed separately for each decision. For example, each variable has its own autocorrelation estimate and effective sample size.

The difference to the binary situation lies in what we do when we are confident enough about the decision of some subset of variables, $V^c \subset V$. Simply stopping all sampling would be too risky, since we are still uncertain about the decisions of $V^u := V \setminus V^c$. Continuing to sample until we are certain about all decision will be wasteful, since we know that variables with marginal close to $0.5$ require many more samples than others for a confident decision. We therefore propose to continue sampling, but only for the variables about which we are still uncertain. This requires us to derive an expression for $p(x_u)$, the marginal probability of all variables that we are still uncertain about.

Computing $p(x_u) = \sum_{\bar{x}_c \in \{0,1\}^{V^c}} p(\bar{x}_c, x_u)$ exactly is almost always infeasible, otherwise, we would not have needed to resort to sampling based inference in the first place. An alternative idea would be to continue using the original factor graph, but to clamp all variables we are certain about to their MMP values. This is computationally feasible, but it results in samples from a conditional distribution, $p(x_u|x_c = z_c)$, not from the desired marginal one. The new construction that we introduce combines advantages of both previous ideas: it is computationally as efficient as the value clamping, but it uses a distribution that approximates the marginal distribution as closely as possible. Similar as in mean-field methods [7], the main step consists of finding distributions $q$ and $q'$ such that $p(x) \approx q(x_u)q'(x_c)$. Subsequently, $q(x_u)$ can be used as approximate replacement to $p(x_u)$, because $p(x_u) = \sum_{\bar{x}_c \in \{0,1\}^{V^c}} p(x) \approx \sum_{\bar{x}_c \in \{0,1\}^{V^c}} q'(\bar{x}_c)q(x_u) = q(x_u)$. The main difference to mean-field inference lies in the fact that $q$ and $q'$ have different role in our construction. For $q'$ we prefer a distribution that factorizes over the variables that we are confident about. Because we want $q$ also to respect the marginal probabilities, $\hat{\mu}_i$ for $i \in V^c$, as estimated them from the sampling process so far, we obtain $q'(x_c) = \prod_{i \in V^c} \hat{\mu}_i^{x_i}(1 - \hat{\mu}_i)^{x_i}$. The distribution $q$ contain all variables that we are not yet confident about, so we want to avoid making any limiting assumptions about its potential values or structure. Instead, we define it as the solution of minimizing $\mathrm{KL}(p|qq')$ over all distributions $q$, which yields the solution

$$q(x_u) \propto \exp\left( -\mathbb{E}_{\bar{x}_c \sim q'(x_c)}\{E(\bar{x}_c, x_u)\} \right). \tag{6}$$

What remains is to define factors $\mathcal{F}'$ and log-potentials $\psi'$, such that $q(x_u) \propto \exp\left( -\sum_{F \in \mathcal{F}'} \psi'_F(x_F) \right)$ while also allowing for efficient sampling from $q$. For this we partition the original factor set into three disjoint sets, $\mathcal{F} = \mathcal{F}^c \cup \mathcal{F}^u \cup \mathcal{F}_0$, with $\mathcal{F}^c := \{F \subset \mathcal{F} : F \subset V^c\}$, $\mathcal{F}^u := \{F \subset \mathcal{F} : F \subset V^u\}$, and $\mathcal{F}_0 := \mathcal{F} \setminus (\mathcal{F}^c \cup \mathcal{F}^u)$. Each factor $F_0 \in \mathcal{F}_0$ we split further into its certain and uncertain components, $F_0^c \subset V^c$ and $F_0^u \subset V^u$, respectively.

With this we obtain a decomposition of the exponent in Equation (6):

$$\mathbb{E}_{\bar{x}_c \sim q'}\{E(\bar{x}_c, x_u)\} = \sum_{F^c \in \mathcal{F}^c} \sum_{\bar{x}_{F^c}} q'(\bar{x}_c)\psi_{F^c}(x_{F^c}) + \sum_{F_u \in \mathcal{F}^u} \psi_{F_u}(x_{F_u}) + \sum_{F_0 \in \mathcal{F}_0} \sum_{\bar{x}_{F_0^c}} q'(\bar{x}_{F_0^c})\psi_F(\bar{x}_{F_0^c}, x_{F_0^u})$$

The first sum is a constant with respect to $x_u$, so we can disregard it in the construction of $\mathcal{F}'$. The factors and log-potentials in the second sum already depend only on $V^u$, so we can re-use them in unmodified form for $\mathcal{F}'$, we set $\psi'_F = \psi_F$ for every $F \in \mathcal{F}^u$. The third sum we rewrite as $\sum_{\{F^u = F \cap V^u : F \in \mathcal{F}_0\}} \psi'_{F^u}(x_{F^u})$, with

$$\psi'_{F^u}(x_u) := \sum_{\bar{x}_c \in \{0,1\}^{F_c}} \left[ \prod_{i \in F_c} \hat{\mu}_i^{\bar{x}_i}(1 - \hat{\mu}_i)^{1-\bar{x}_i} \right] \psi_F(\bar{x}_c, x_u). \tag{7}$$

for any $F \in \mathcal{F}_0$, where we have made use of the explicit form of $\bar{q}$. If factors with identical variable set occur during this construction, we merge them by summing their log-potentials. Ultimately, we obtain a new factor set $\mathcal{F}' := \mathcal{F}^u \cup \{F \cap V^u : F \in \mathcal{F}_0\}$, and probability distribution

$$q(x_u) \propto \exp\left( \sum_{F \in \mathcal{F}'} \psi'_F(x_F) \right) \qquad \text{for } x_u \in \{0,1\}^{V^u}. \tag{8}$$

Note that during the process, not only the number of variables is reduced, but also the number of factors and the size of each factor can never grow. Consequently, if sampling was feasible for the original distribution $p$, it will also be feasible for $q$, and potentially more efficient.

## 3   Related Work

Sequential sampling with the option of early stopping has a long tradition in Bayesian statistics. First introduced by Wald in 1945 [18], the ability to continuously accumulate information until a decision can be made with sufficient confidence was one of the key factors that contributed to the success of

Bayesian reasoning for decision making. Today, it has been a standard technique in areas as diverse as clinical medicine (e.g. for early stop of drug trials [19]), social sciences (e.g. for designing and evaluating experiments [20]), and economics (e.g. in modelling stock market behavior [21]).

In current machine learning research, sequential sampling is used less frequently for making individual decisions, but in the form of MCMC it has become one of the most successful techniques for statistical inference of probability distributions with many dependent variables [12, 22]. Nevertheless, to the best of our knowledge, the method we propose is the first one that performs early stopping of subsets of variables in this context. Many other approaches to reduce the complexity of sampling iterations exist, however, for example to approximate complex graphical models by simpler ones, such as trees [23], or loopy models of low treewidth [24]. These fall into a different category than the proposed method, though, as they are usually performed statically and prior to the actual inference step, so they cannot dynamically assign computational resources where they are needed most. *Beam search* [25] and related techniques take an orthogonal approach to ours. They dynamically exclude low-likelihood label combinations from the inference process, but they keep the size and topology of the factor graph fixed. *Select and sample* [26] disregards a data-dependent subset of variables during each sampling iterations. It is not directly applicable in our situation, though, since it requires that the underlying graphical model is bipartite, such that the individual variables are conditionally independent of each other. Given their complementary nature, we believe that the idea of combining adaptive MMP with *beam search* and/or *select and sample* could be a promising direction for future work.

## 4 Experimental Evaluation

To demonstrate the effect of adaptive MMP compared to naive MMP, we performed experiments in two prototypical applications: *multi-label classification* and *binary image inpainting*. In both tasks, performance is typically measured by the Hamming loss, so MMP is the preferred method of test time prediction.

### 4.1 Multi-Label Classification

In multi-label classification, the task is to predict for each input $y \in \mathcal{Y}$, which labels out of a label set $\mathcal{L} = \{1, \ldots, K\}$ are correct. The difference to multi-class classification is that several labels can be correct simultaneously, or potentially none at all. Multi-label classification can be formulated as simultaneous prediction of $K$ binary labels $(x_i)_{i=1,\ldots K}$, where $x_i = 1$ indicates that the label $i$ is part of the prediction, and $x_i = 0$ indicates that it is not. Even though multi-label classification can in principle be solved by training $K$ independent predictors, several studies have shown that by making use of dependencies between label, the accuracy of the individual predictions can be improved [1, 27, 28].

For our experiments we follow [1] in using a fully-connected conditional random field model. Given an input $y$, each label variable $i$ has a unary factor $F_i = \{i\}$ with log-linear potential $\psi_i(x_i) = \langle w_i, y \rangle x_i$, where $w_i$ is a label-specific weight vector that was learned from training data. Additionally there are $K(K-1)/2$ pairwise factors, $F_{ij} = \{i, j\}$, with log-potentials $\psi_{ij}(x_i, x_j) = \eta_{ij} x_i x_j$. Its free parameter $\eta_{ij}$ is learned as well. The resulting conditional joint distribution has the form of a Boltzmann machine, $p(x|y) \propto \exp(-E_y(x))$, with energy function $E_y(x) = \sum_{i=1}^K \eta_i x_i + \sum_{i=1}^L \sum_{j=i+1}^L \eta_{ij} x_i x_j$ in minimal representation, where $\eta_i$ and $\eta_{ij}$ depend on $y$. We downloaded several standard datasets and trained the CRF on each of them using a stochastic gradient descent procedure based on the `sgd`[2] package. The necessary gradients are computing using a junction tree algorithms for problems with 20 variables or less, and by Gibbs sampling otherwise. For model selection, when required, we used 10-fold cross-validation on the training set.

Note that our goal in this experiment is not to advocate a new model multi-label classification, but to create probability distributions as they would appear in real problems. Nevertheless, we also report classification accuracy in Table 1 to show that a) the learned models have similar characteristics as earlier work, in particular to [29], where the an identical model was trained using structured SVM learning, and b) adaptive MMP can achieve as high prediction accuracy as ordinary Gibbs sampling, as long as the confidence parameter $\epsilon$ is not chosen overly optimistically. In fact, in many cases even

| Dataset | #Labels | #Train | #Test | [29] | [28] | Exact | Gibbs | Proposed |
|---|---|---|---|---|---|---|---|---|
| Synth1 [29] | 6 | 471 | 5045 | 6.9 | — | 5.2 | 5.3 | 5.2 / 5.2 / 5.2 |
| Synth2 [29] | 10 | 1000 | 10000 | 7.0 | — | 10.0 | 10.0 | 10.0/10.0/10.0 |
| Scene | 6 | 1211 | 1196 | 10.1 | $9.5 \pm 2.1$ | 10.4 | 10.3 | 10.2/10.2/10.2 |
| Rcv1-10 [29] | 10 | 2916 | 2914 | 5.6 | — | 4.2 | 4.2 | 4.6 / 4.4 / 4.2 |
| Mediamill-10 [29] | 10 | 29415 | 12168 | 18.8 | — | 18.4 | 18.6 | 19.0/18.6/18.4 |
| Yeast | 14 | 1500 | 917 | 20.2 | $20.2 \pm 1.3$ | 20.0 | 20.2 | 23.4/21.4/20.5 |
| Tmc2007 | 22 | 21519 | 7077 | — | $3.3 \pm 2.7$ | 5.3 | 5.3 | 5.3 / 5.3 / 5.3 |
| AwA [30] | 85 | 24295 | 6180 | — | — | — | 32.2 | 32.7/32.7/32.7 |
| Mediamill | 101 | 29415 | 12168 | — | $3.6 \pm 0.5$ | — | 3.7 | 3.6 / 3.5 / 3.6 |
| Rcv1 | 103 | 3000 | 3000 | — | — | — | 1.5 | 1.7 / 1.6 / 1.5 |

Table 1: Multi-label classification. Dataset characteristics (number of labels, number of training examples, number of test examples) and classification error rate in percent. [29] used the same model as we do, but trained it using a structured SVM framework and predicted using MAP. [28] compared 12 different multi-label classification techniques, we report their mean and standard deviation. The remaining columns give MMP prediction accuracy of the trained CRF models: *Exact* computes the exact marginal values by a junction tree, *Gibbs* and *Proposed* performs ordinary Gibbs sampling, or the proposed adaptive version with $\epsilon = 10^{-2}/10^{-5}/10^{-8}$, both run for up to 500 iterations.

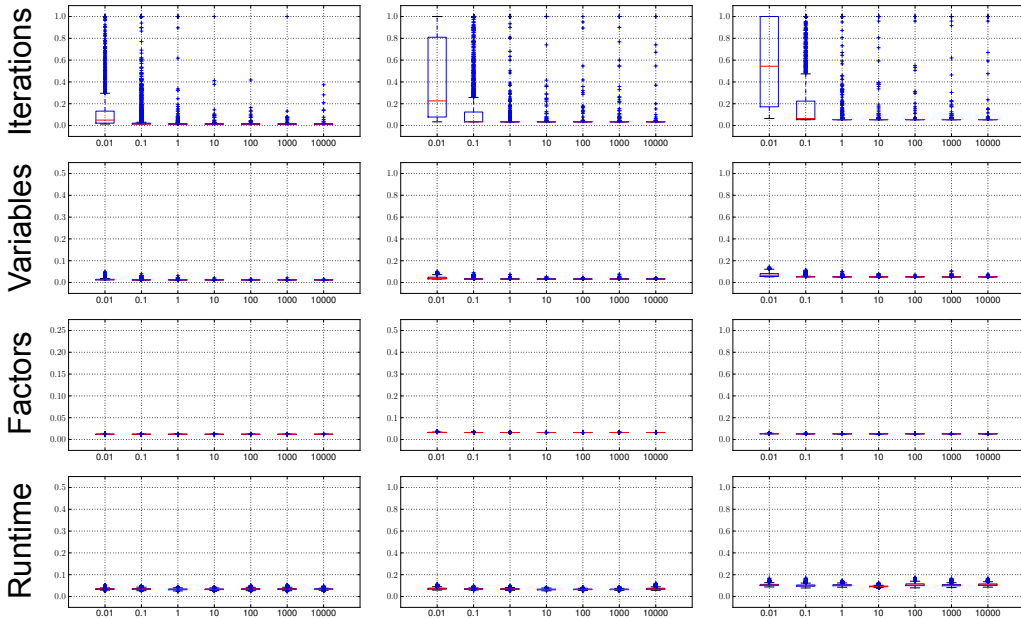

Figure 1: Results of adaptive pruning on RCV1 dataset for $\epsilon = 10^{-2}, 10^{-5}, 10^{-8}$ (left to right). $x$-axis: regularization parameter $C$ used for training, $y$-axis: ratio of iterations/variables/factors/ runtime used by adaptive sampling relative to Gibbs sampling.

a relative large value, such as $\epsilon = 0.01$ results in a smaller loss of accuracy than the potential 1%, but overall, a value of $10^{-5}$ or less seems advisable.

Figures 1 and 2 show in more detail how the adaptive sampling behaves on two exemplary datasets with respect to four aspects: the number of *iterations*, the number of *variables*, the number of *factors*, and the *overall runtime*. For each aspect we show a box plot of the corresponding relative quantity compared to the Gibbs sampler. For example, a value of 0.5 in *iterations* means that the adaptive sample terminated after 250 iterations instead of the maximum of 500, because it was confident about all decisions. Values of 0.2 in *variables* and *factors* means that the number of variable states samples by the adaptive sampler was 20%, and the number of factors in the corresponding factor graphs was 10% of the corresponding quantities for the Gibbs sampler. Within each plot, we reported results for the complete range of regularization parameters in order to illustrate the effect that regularization has on the distribution of marginals.

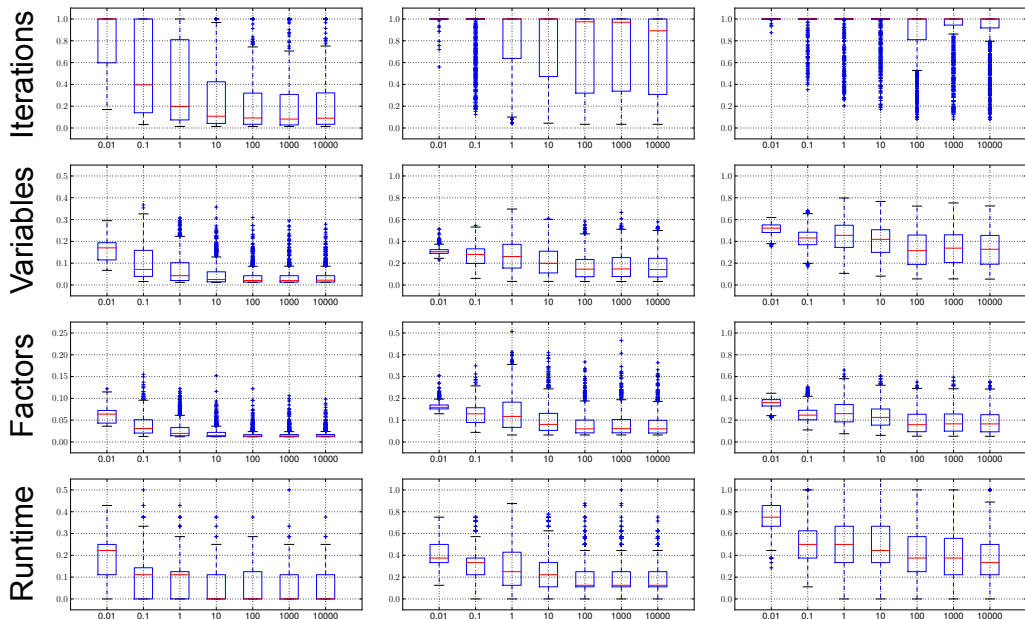

Figure 2: Results of adaptive pruning on YEAST dataset for $\epsilon = 10^{-2}, 10^{-5}, 10^{-8}$ (left to right). $x$-axis: regularization parameter $C$ used for training, $y$-axis: ratio of iterations/variables/factors/ runtime used by adaptive sampling relative to Gibbs sampling. Note that the scaling of the $y$-axis differs between columns.

Figure 1 shows results for the relatively simple RCV1 dataset. As one can see, a large number of variables and factors are removed quickly from the factor graph, leading to a large speedup compared to the ordinary Gibbs sampler. In fact, as the first row shows, it was possible to make a confident decision for all variables far before the 500th iteration, such that the adaptive method terminated early. As a general trend, the weaker the regularization (larger $C$ value in the plot), the earlier the adaptive sampler is able to remove variables and factors, presumably because more extreme values of the energy function result in more marginal probabilities close to 0 or 1. A second insight is that despite the exponential scaling of the confidence parameter between the columns, the runtime grows only roughly linearly. This indicates that we can choose $\epsilon$ conservatively without taking a large performance hit. On the hard YEAST dataset (Figure 2) in the majority of cases the adaptive sampling does not terminate early, indicating that some of the variables have marginal probabilities close to $0.5$. Nevertheless, a clear gain in speed can be observed, in particular in the weakly regularized case, indicating that nevertheless, many tests for confidence are successful early during the sampling.

## 4.2 Binary Image Inpainting

Inpainting is a classical image processing task: given an image (in our case black-and-white) in which some of the pixels are occluded or have missing values, the goal is to predict a completed image in which the missing pixels are set to their correct value, or at least in a visually pleasing way. Image inpainting has been tackled successfully by grid-shaped Markov random field models, where each pixel is represented by a random variable, unary factors encode local evidence extracted from the image, and pairwise terms encode the cooccurrence of pixel value. For our experiment, we use the *Hard Energies from Chinese Characters (HECC)* dataset [31], for which the authors provide pre-computed energy functions. The dataset has 100 images, each with between 4992 and 17856 pixels, i.e. binary variables. Each variable has one unary and up to 64 pairwise factors, leading to an overall factor count of 146224 to 553726. Because many of the pairwise factors act repulsively, the underlying energy function is highly non-submodular, and sampling has proven a more successful mean of inference than, for example, message passing [31].

Figure 3 shows exemplary results of the task. The complete set can be found in the supplemental material. In each case, we ran an ordinary Gibbs sampler and the adaptive sampler for 30 seconds,

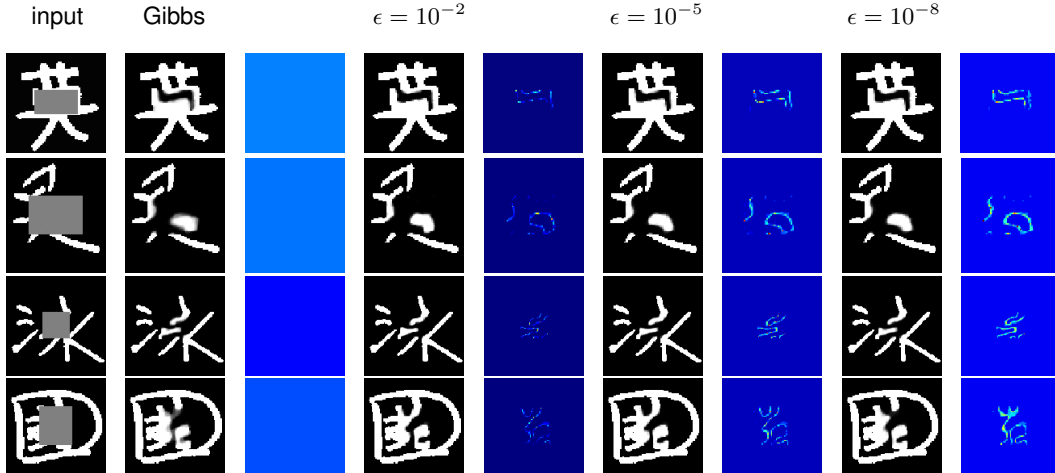

Figure 3: Example results of binary image inpainting on *HECC* dataset. From left to right: image to be inpainted, result of Gibbs sampling, result of adaptive sampling, where each method was run for up to 30 seconds per image. The left plot of each result shows the marginal probabilities, the right plot shows how often each pixel was sampled on a log scale from 10 (dark blue) to 100000 (bright red). Gibbs sampling treats all pixels uniformly, reaching around 100 sampling sweeps within the given time budget. Adaptive sampling stops early for parts of the image that it is certain about, and concentrates its samples in the uncertain regions, i.e. pixels with marginal probability close to 0.5. The larger $\epsilon$, the more pronounced this effect it.

and we visualize the resulting marginal probabilities as well as the number of samples created for each of the pixels. One can see that adaptive sampling comes to a more confident prediction within the given time budget. The larger the $\epsilon$ parameter, the earlier to stops sampling the 'easy' pixels, spending more time on the difficult cases, i.e. pixel with marginal probability close to 0.5.

## 5  Summary and Outlook

In this paper we derived an analytic expression for how confident one can be about the maximum marginal predictions (MMPs) of a binary graphical model after a certain number of samples, and we presented a method for pruning factor graphs when we want to stop sampling for a subset of the variables. In combination, this allowed us to more efficiently infer the MMPs: starting from the whole factor graph, we sample sequentially, and whenever we are sufficiently certain about a prediction, we prune it from the factor graph before continuing to sample. Experiments on multi-label classification and image inpainting show a clear increase in performance at virtually no loss in accuracy, unless the confidence is chosen too optimistically.

Despite the promising results there are two main limitations that we plan to address. On the one hand, the multi-label experiments showed that sometimes, a conservative estimate of the confidence is required to achieve highest accuracy. This is likely a consequence of the fact that our pruning uses the estimated marginal to build a new factor graph, and even if the decision confidence is high, the marginals can still vary considerably. We plan to tackle this problem by also integrating bounds on the marginals with data-dependent confidence into our framework. A second limitation is that we can currently only handle binary-valued labelings. This is sufficient for multi-label classification and many problems in image processing, but ultimately, one would hope to derive similar early stopping criteria also for graphical models with larger label set. Our pruning method would be readily applicable to this situation, but an open challenge lies in finding a suitable criterion when to prune variables. This will require a deeper understanding of tail probabilities of multinomial decision variables, but we are confident it will be achievable, for example based on existing prior works from the case of i.i.d. samples [14, 32].

## Footnotes

[1] Many more involved methods for estimating the effective sample size exist, see, for example, [17], but in our experiments the first-order method proved sufficient for our purposes.

[2]http://leon.bottou.org/projects/sgd

# References

[1] N. Ghamrawi and A. McCallum. Collective multi-label classification. In *CIKM*, 2005.

[2] X. Zhu, Z. Ghahramani, and J. Lafferty. Semi-supervised learning using Gaussian fields and harmonic functions. In *ICML*, 2003.

[3] S. Geman and D. Geman. Stochastic relaxation, Gibbs distributions, and the Bayesian restoration of images. *PAMI*, 6(6), 1984.

[4] S. Della Pietra, V. Della Pietra, and J. Lafferty. Inducing features of random fields. *PAMI*, 19(4), 1997.

[5] C. Yanover and Y. Weiss. Approximate inference and protein folding. In *NIPS*, volume 15, 2002.

[6] E. Schneidman, M. J. Berry, R. Segev, and W. Bialek. Weak pairwise correlations imply strongly correlated network states in a neural population. *Nature*, 440(7087), 2006.

[7] M. J. Wainwright and M. I. Jordan. Graphical models, exponential families, and variational inference. *Foundations and Trends in Machine Learning*, 1(1-2), 2008.

[8] J. Marroquin, S. Mitter, and T. Poggio. Probabilistic solution of ill-posed problems in computational vision. *Journal of the American Statistical Association*, 82(397), 1987.

[9] C. Yanover and Y. Weiss. Finding the $m$ most probable configurations using loopy belief propagation. In *NIPS*, volume 16, 2004.

[10] M. Jerrum and A. Sinclair. Polynomial-time approximation algorithms for the Ising model. *SIAM Journal on Computing*, 22, 1993.

[11] R. M. Neal. Probabilistic inference using Markov chain Monte Carlo methods. Technical Report CRG-TR-93-1, Department of Computer Science, University of Toronto, 1993.

[12] D. J. C. MacKay. Introduction to Monte Carlo methods. In *Proceedings of the NATO Advanced Study Institute on Learning in graphical models*, 1998.

[13] A. E. Raftery and S. Lewis. How many iterations in the Gibbs sampler. *Bayesian Statistics*, 4(2), 1992.

[14] A. G. Schwing, C. Zach, Y. Zheng, and M. Pollefeys. Adaptive random forest – how many "experts" to ask before making a decision? In *CVPR*, 2011.

[15] H. Weiler. The use of incomplete beta functions for prior distributions in binomial sampling. *Technometrics*, 1965.

[16] C. M. Thompson, E. S. Pearson, L. J. Comrie, and H. O. Hartley. Tables of percentage points of the incomplete beta-function. *Biometrika*, 1941.

[17] R. V. Lenth. Some practical guidelines for effective sample size determination. *The American Statistician*, 55(3), 2001.

[18] A. Wald. Sequential tests of hypotheses. *Annals of Mathematical Statistics*, 16, 1945.

[19] D. A. Berry. Bayesian clinical trials. *Nature Reviews Drug Discovery*, 5(1), 2006.

[20] A. E. Raftery. Bayesian model selection in social research. *Sociological Methodology*, 25, 1995.

[21] D. Easley, N. M. Kiefer, M. O'hara, and J. B. Paperman. Liquidity, information, and infrequently traded stocks. *Journal of Finance*, 1996.

[22] C. J. Geyer. Practical Markov chain Monte Carlo. *Statistical Science*, 7(4), 1992.

[23] C. Chow and C. Liu. Approximating discrete probability distributions with dependence trees. *IEEE Transactions on Information Theory*, 14(3), 1968.

[24] F. Bach and M. I. Jordan. Thin junction trees. In *NIPS*, volume 14, 2002.

[25] M. J. Collins. A new statistical parser based on bigram lexical dependencies. In *ACL*, 1996.

[26] J. A. Shelton, J. Bornschein, A. S. Sheikh, P. Berkes, and J. Lücke. Select and sample – a model of efficient neural inference and learning. In *NIPS*, volume 24, 2011.

[27] Y. Guo and S. Gu. Multi-label classification using conditional dependency networks. In *IJCAI*, 2011.

[28] G. Madjarov, D. Kocev, D. Gjorgjevikj, and S. Dzeroski. An extensive experimental comparison of methods for multi-label learning. *Pattern Recognition*, 2012.

[29] T. Finley and T. Joachims. Training structural SVMs when exact inference is intractable. In *ICML*, 2008.

[30] C. H. Lampert, H. Nickisch, and S. Harmeling. Learning to detect unseen object classes by between-class attribute transfer. In *CVPR*, 2009.

[31] S. Nowozin, C. Rother, S. Bagon, T. Sharp, B. Yao, and P. Kohli. Decision tree fields. In *ICCV*, 2011.

[32] D. Chafai and D. Concordet. Confidence regions for the multinomial parameter with small sample size. *Journal of the American Statistical Association*, 104(487), 2009.

